# Learning efficient auditory codes using spikes predicts cochlear filters

**Evan Smith**[1]          **Michael S. Lewicki**[2]
**evan@cnbc.cmu.edu  lewicki@cnbc.cmu.edu**

Departments of Psychology[1] & Computer Science[2]
Center for the Neural Basis of Cognition
Carnegie Mellon University

## Abstract

The representation of acoustic signals at the cochlear nerve must serve a wide range of auditory tasks that require exquisite sensitivity in both time and frequency. Lewicki (2002) demonstrated that many of the filtering properties of the cochlea could be explained in terms of efficient coding of natural sounds. This model, however, did not account for properties such as phase-locking or how sound could be encoded in terms of action potentials. Here, we extend this theoretical approach with algorithm for learning efficient auditory codes using a spiking population code. Here, we propose an algorithm for learning efficient auditory codes using a theoretical model for coding sound in terms of spikes. In this model, each spike encodes the precise time position and magnitude of a localized, time varying kernel function. By adapting the kernel functions to the statistics natural sounds, we show that, compared to conventional signal representations, the spike code achieves far greater coding efficiency. Furthermore, the inferred kernels show both striking similarities to measured cochlear filters and a similar bandwidth versus frequency dependence.

## 1   Introduction

Biological auditory systems perform tasks that require exceptional sensitivity to both spectral and temporal acoustic structure. This precision is all the more remarkable considering these computations begin with an auditory code that consists of action potentials whose duration is in milliseconds and whose firing in response to hair cell motion is probabilistic. In computational audition, representing the acoustic signal is the first step in any algorithm, and there are numerous approaches to this problem which differ in both their computational complexity and in what aspects of signal structure are extracted. The auditory nerve representation subserves a wide variety of different auditory tasks and is presumably well-adapted for these purposes. Here, we investigate the theoretical question of what computational principles might underlie cochlear processing and the representation of the auditory nerve.

For sensory representations, a theoretical principle that has attracted considerable interest is efficient coding. This posits that (assuming low noise) one goal of sensory coding

is to represent signals in the natural sensory environment efficiently, i.e. with minimal redundancy [1–3]. Recently, it was shown that efficient coding of natural sounds could explain auditory nerve filtering properties and their organization as a population [4] and also account for some non-linear properties of auditory nerve responses [5]. Although those results provided an explanation for auditory nerve encoding of spectral information, they fail to explain the encoding of temporal information. Here, we extend the standard efficient coding model, which has an implicit stationarity assumption, to form efficient representations of non-stationary and time-relative acoustic structures.

## 2 An abstract model for auditory coding

In standard models of efficient coding, sensory signals are represented by vectors of fixed length, and the representation is a linear transformation of the input pattern. A simple method to encode temporal signals is to divide the signal into discrete blocks; however, this approach has several drawbacks. First, the underlying acoustic structures have no relation to the block boundaries, so elemental acoustic features may be split across blocks. Second, this representation implicitly assumes that the signal structures are stationary, and provides no way to represent *time-relative* structures such as transient sounds. Finally, this approach has limited plausibility as a model of cochlear encoding. To address all of these problems, we use a theoretical model in which sounds are represented as spikes [6,7]. In this model, the signal, $x(t)$, is encoded with a set of kernel functions, $\phi_1 \dots \phi_M$, that can be positioned arbitrarily and independently in time. The mathematical form of the representation with additive noise is

$$x(t) = \sum_{m=1}^{M} \sum_{i=1}^{n_m} s_i^m \, \phi_m(t - \tau_i^m) + \epsilon(t), \tag{1}$$

where $\tau_i^m$ and $s_i^m$ are the temporal position and coefficient of the $i^{\text{th}}$ instance of kernel $\phi_m$, respectively. The notation $n_m$ indicates the number of instances of $\phi_m$, which need not be the same across kernels. In addition, the kernels are not restricted in form or length.

The key theoretical abstraction of the model is that the signal is decomposed in terms of discrete acoustic events, each of which has a precise amplitude and temporal position. We interpret the analog amplitude values as representing a local population of auditory nerve spikes. Thus, this theory posits that the purpose of the (binary) spikes at the auditory nerve is to encode as accurately as possible the temporal position and amplitude of the acoustic events defined by $\phi_m(t)$. The main questions we address are 1) *encoding*, i.e. what are the optimal values of $\tau_i^m$ and $s_i^m$ and 2) *learning*, i.e. what are the optimal kernel functions $\phi_m(t)$.

### 2.1 Encoding

Finding the optimal representation of arbitrary signals in terms of spikes is a hard problem, and currently there are no known biologically plausible algorithms that solve this problem well [7]. There are reasons to believe that this problem can be solved (approximately) with biological mechanisms, but for our purposes here, we compute the values of $\tau_i^m$ and $s_i^m$ for a given signal we using the matching pursuit algorithm [8]. It iteratively approximates the input signal with successive orthogonal projections onto a basis. The signal can be decomposed into

$$x(t) = < x(t)\phi_m > \phi_m + R_x(t), \tag{2}$$

where $< x(t)\phi_m >$ is the inner product between the signal and the kernel and is equivalent to $s_i^m$ in equation 1. The final term in equation 2, $R_x(t)$, is the residual signal after approximating $x(t)$ in the direction of $\phi_m$. The projection with the largest magnitude inner product will minimize the power of $R_x(t)$, thereby capturing the most structure possible *with a single kernel*.

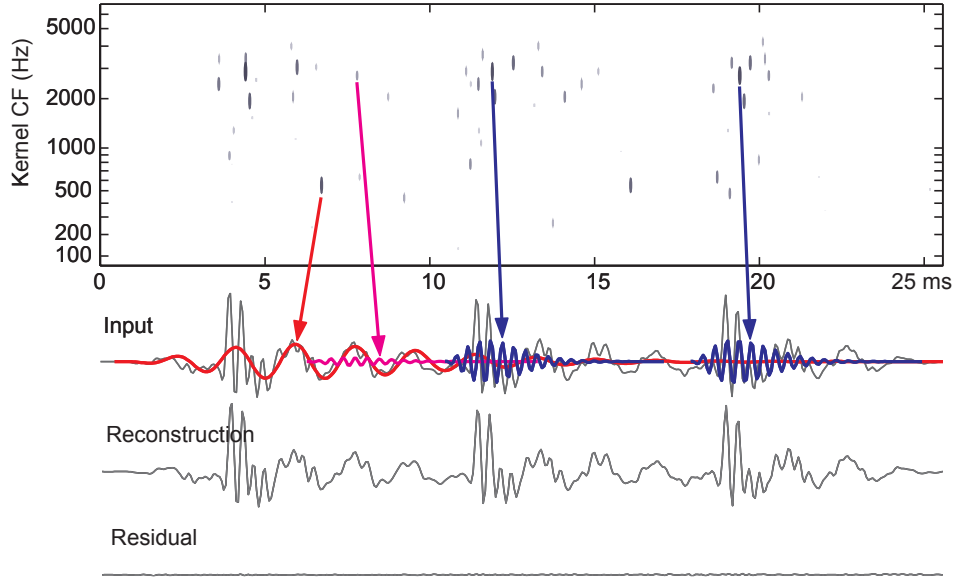

Figure 1: A brief segment of the word canteen (input) is represented as a spike code (top). A reconstruction of the speech based only on the few spikes shown (ovals in spike code) is very accurate with relatively little residual error (reconstruction and residual). The colored arrows and matching curves illustrate the correspondence between a few of the ovals and the underlying acoustic structure represented by the kernel functions.

Equation 2 can be rewritten more generally as

$$R_x^n(t) = < R_x^n(t)\phi_m > \phi_m + R_x^{n+1}(t), \tag{3}$$

with $R_x^0(t) = x(t)$ at the start of the algorithm. On each iteration, the current residual is projected onto the basis. The projection with the largest inner product is subtracted out, and its coefficient and time are recorded. This projection and subtraction leaves $< R_x^n(t)\phi_m > \phi_m$ orthogonal to the residual signal, $R_x^{n+1}(t)$ and to all previous and future projections [8]. As a result, matching pursuit codes are composed of mutually orthogonal signal structures. For the results reported here, the encoding was halted when $s_i^m$ fell below a preset threshold (the spiking threshold).

Figure 1 illustrates the spike code model and its efficiency in representing speech. The spoken word "canteen" was encoded as a set of spikes using a fixed set of kernel functions. The kernels can have arbitrary shape and for illustration we have chosen gammatones (mathematical approximations of cochlear filters) as the kernel functions. A brief segment from input signal (1, Input) consists of three glottal pulses in the /a/ vowel. The resulting spike code is show above it. Each oval represents the temporal position and center frequency of an underlying kernel function, with oval size and gray value indicating kernel amplitude. For four spikes, colored arrows and curves indicate the relationship between the ovals and the acoustics events they represent. As evidenced from the figure, the very small set of spike events is sufficient to produce a very accurate reconstruction of the sound (reconstruction and residual).

## 2.2 Learning

We adapt the method used in [9] to train our kernel function. Equation 1 can be rewritten in probabilistic form as

$$p(x|\Phi) = \int p(x|\Phi, \hat{s}) p(\hat{s}) ds, \tag{4}$$

where $\hat{s}$, an approximation of the posterior maximum, comes from the set of coefficient generated by matching pursuit. We assume the noise in the likelihood, $p(x|\Phi, \hat{s})$, is Gaussian and the prior, $p(s)$, is sparse. The basis is updated by taking the gradient of the log probability,

$$\frac{\partial}{\partial \phi_m} \log(p(x|\Phi)) \quad = \quad \frac{\partial}{\partial \phi_m} \log(p(x|\Phi, s)) + \log(p(s)) \tag{5}$$

$$= \quad \frac{1}{2\sigma_\varepsilon} \frac{\partial}{\partial \phi_m} [x - \sum_{m=1}^{M} \sum_{i=1}^{n_m} \hat{s}_i^m \phi_m(t - \tau_i^m)]^2 \tag{6}$$

$$= \quad \frac{1}{\sigma_\varepsilon} [x - \hat{x}] \sum_i \hat{s}_i^m \tag{7}$$

As noted by Olshausen (2002), equation 7 indicates that the kernels are updated in Hebbian fashion, simply as a product of activity and residual [9] (i.e., the unit shifts its preferred stimuli in the direction of the stimuli that just made it spike minus those elements already encoded by other units). But in the case of the spike code, rather than updating for every time-point, we need only update at times when the kernel spiked.

As noted earlier, the model can use kernels of any form or length. This capability also extends to the learning algorithm such that it can learn functions of differing temporal extents, growing or shrinking them as needed. Low frequency functions and others requiring longer temporal extent can be grown from shorter initial seeds, while brief functions can be trimmed to speed processing and minimize the effects of over-fitting. Periodically during training, a simple heuristic is used to trim or extend the kernels, $\phi_m$. The functions are initially zero-padded. If learning causes the power of the padding to surpass a threshold, the padding is extended. If the power of the padding plus an adjacent segment falls below the threshold, the padding is trimmed from the end. Following the gradient step and length adjustment, the kernels are again normalized and the next training signal is encoded.

## 3 Adapting kernels to natural sounds

The spike coding algorithm was used to learn kernel functions for two different classes of sounds: human speech and music. For speech, the algorithm trained on a subset the TIMIT Speech Corpus. Each training sample consisted of a single speaker saying a single sentence. The signals were bandpass filtered to remove DC components of the signal and to prevent aliasing from affecting learning. The signals were all normalized to a maximum amplitude of 1.

Each of the 30 kernel functions were initialized to random Gaussian vectors of 100 samples in duration. The threshold below which spikes (values of $s_m$) were ignored during the encoding stage was set at 0.1, which allowed for an initial encoding of $\sim$ 12dB signal-to-noise ratio (SNR). As indicated by equation 7, the gradient depends on the residual. If the residual drops near zero or is predominately noise then learning is impeded. By slowly increasing the spiking threshold as the average residual drops, we retain some signal structure in the residual for further training. At the same time, the power distribution of natural sounds means that high frequency signal components might fall entirely below threshold, preventing their being learned. One possible solution that was not implemented here is using separate thresholds for each kernel.

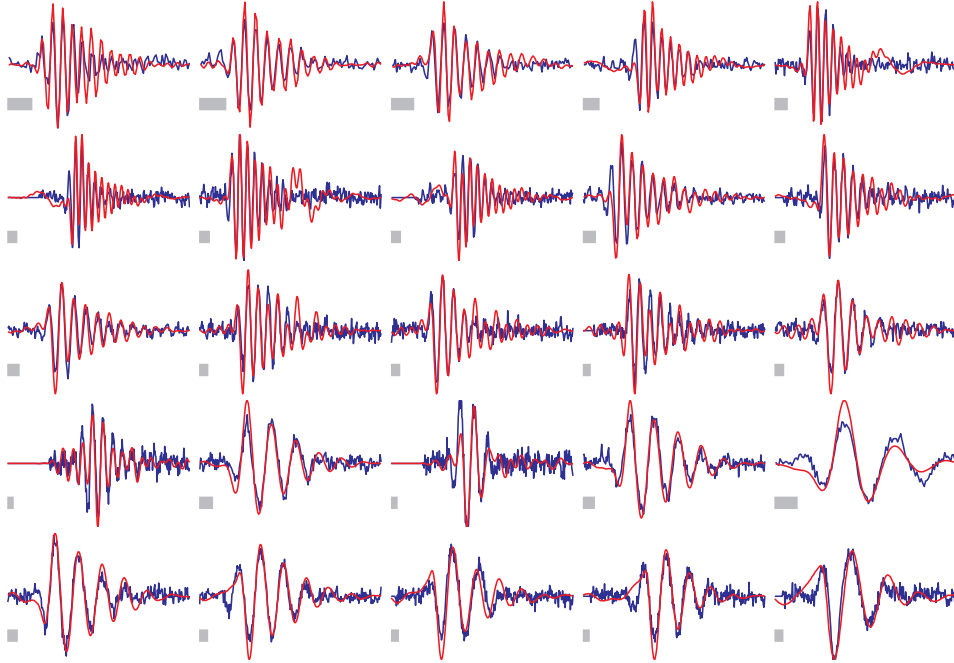

Figure 2: When adapted to speech, kernel functions become asymmetric sinusoids (smooth curves in red, zero padding has been removed for plotting), with sharp attacks and gradual decays. They also adapt in temporal extent, with longer and shorter functions emerging from the same initial length. These learned kernels are strikingly similar to revcor functions obtained from cat auditory nerve fibers (noisy curves in blue). The revcor functions were normalized and aligned in phase with the learned kernels but are otherwise unaltered (no smoothing or fitting).

Figure 2 shows the kernel functions trained on speech (red curves). All are temporally localized, bandpass filters. They are similar in form to previous results but with several notable differences. Most notably, the learned kernel functions are temporally asymmetric, with sharp attack and gradual decay which matches physiological filtering properties of the auditory nerves. Each kernel function in figure 2 is overlayed on a so-called reverse-correlation (revcor) function which is an estimate of the physiological impulse response function for an individual auditory nerve fiber [10]. The revcor functions have been normalized, and the most closely matching in terms of center frequency and envelop were phase aligned with learned kernels by hand. No additional fitting was done, yet there is a striking similarity between the inferred kernels functions and physiologically estimated reverse-correlation functions. For 25 out of 30 kernel functions, we found a close match to the physiological revcor functions (correlation > 0.8). Of the remaining filters, all possessed the same basic asymmetric filter structure show in figure 2 and showed a more modest match to the data (correlation > 0.5).

In the standard efficient coding model, the signal and the basis functions are all the same length. In order for the basis to span the signal space in the time domain and still be temporally localized, some of the learned functions are essentially replications of one another. In the spike coding model, this redundancy does not occur because coding is time-relative. Kernel functions can be placed arbitrarily in time such that one kernel function can code for similar acoustic events at different points in the signal. So, temporally extended functions can be learned without causing an explosion in the number of high-frequency functions

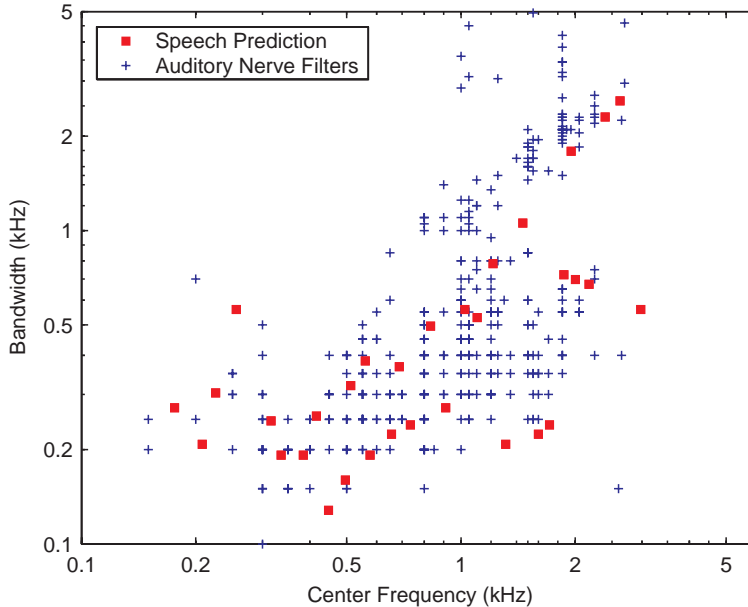

Figure 3: The center frequency vs. bandwidth distribution of learned kernel functions (red squares) plotted against physiological data (blue pluses).

needed to span the signal space. Because cochlear coding also shares this quality, it might also allow more precise predictions about the population characteristics of cochlear filters.

Individually, the learned kernel functions closely match the linear component of cochlear filters. We can also compare the learned kernels against physiological data in terms of population distributions. In frequency space, our learned population follows the approximately logarithmic distribution found in the cochlea, a more natural distribution of filters compared to previous findings, where the need to tile high-frequency space biased the distribution [4]. Figure 3 presents a log-log scatter-plot of the center frequency of each kernel versus its bandwidth (red squares). Plotted on the same axis are two sets of empirical data. One set (blue pluses) comes from a large corpus of reverse-correlation functions derived from physiological recordings of auditory nerve fibers [10]. Both the slope and distribution of the learned kernel functions match those of the empirical data. The distribution of learned kernels even appears to follow shifts in the slope of the empirical data at the high and low frequencies.

## 4   Coding Efficiency

We can quantify the coding efficiency of the learned kernel functions in bits so as to objectively evaluate the model and compare it quantitatively to other signal representations. Rate-fidelity provides a useful objective measure for comparison. Here we use a method developed in [7] which we now briefly describe. Computing the rate-fidelity curves begins with associated pairs of coefficients and time values, $\{s_i^m, \tau_i^m\}$, which are initially stored as double precision variables. Storing the original time values referenced to the start of the signal is costly because their range can be arbitrarily large and the distribution of time points is essentially uniform. Storing only the time since the last spike, $\delta\tau_i^m$, greatly restricts the range and produces a variable that approximately follows a gamma distribution.

Rate-fidelity curves are generated by varying the precision of the code, $\{s_i^m, \delta\tau_i^m\}$, and computing the resulting fidelity through reconstruction. A uniform quantizer is used to

vary the precision of the code between 1 and 16 bits. At all levels of precision, the bin widths for quantization are selected so that equal numbers of values fall in each bin. All $s_i^m$ or $\delta\tau_i^m$ that fall within a bin are recoded to have the same value. We use the mean of the non-quantized values that fell within the bin. $s_i^m$ and $\delta\tau_i^m$ are quantized independently.

Treating the quantized values as samples from a random variable, we estimate a code's entropy (bits/coefficient) from histograms of the values. Rate is then the product of the estimated entropy of the quantized variables and the number of coefficients per second for a given signal. At each level of precision the signal is reconstructed based on the quantized values, and an SNR for the code is computed. This process was repeated across a set of signals and the results were averaged to produce rate-fidelity curves.

Coding efficiency can be measured in nearly identical fashion for other signal representations. For comparison we generate rate-fidelity curves for Fourier and wavelet representations as well as for a spike code using either learned kernel functions or gammatone functions. Fourier coefficients were obtained for each signal via Fast Fourier Transform. The real and imaginary parts were quantized independently, and the rate was based on the estimated entropy of the quantized coefficients. Reconstruction was simply the inverse Fourier transform of the quantized coefficients. Similarly, coding efficiency using Daubechies wavelets was estimated using Matlab's discrete wavelet transform and inverse wavelet transform functions. Curves for the gammatone spike code were generated as described above.

Figure 4 shows the rate-fidelity curves calculated for speech from the TIMIT speech corpus [11]. At low bit rates (below 40 Kbps), both of the spike codes produce more efficient representations of speech than the other traditional representations. For example, between 10 and 20 Kbps the fidelity of the spike representation of speech using learned kernels is approximately twice that of either Fourier or wavelets. The learned kernels are also sightly but significantly more efficient than spike codes using gammatones, particularly in the case of music. The kernel functions trained on music are more extended in time and appear better able to describe harmonic structure than the gammatones. As the number of spikes increases the spike codes become less efficient, with the curve for learned kernels dropping more rapidly than for gammatones. Encoding sounds to very high precision requires setting the spike threshold well below the threshold used in training. It may be that the learned kernel functions are not well adapted to the statistics of very low amplitude sounds. At higher bit rates (above 60 Kbps) the Fourier and wavelet representations produce much higher rate-fidelity curves than either spike codes.

## 5   Conclusion

We have presented a theoretical model of auditory coding in which temporal kernels are the elemental features of natural sounds. The essential property of these features is that they can describe acoustic structure at arbitrary time points, and can thus represent non-stationary, transient sounds in a compact and shift-invariant manner. We have shown that by using this time-relative spike coding model and adapting the kernel shapes to efficiently code natural sounds, it is possible to account for both the detailed filter shapes of auditory nerve fibers and their distribution as a population. Moreover, we have demonstrated quantitatively that, at a broad range of low to medium bit rates, this type of code is substantially more efficient than conventional signal representations such as Fourier or wavelet transforms.

## References

[1]  H. B. Barlow. Possible principles underlying the transformation of sensory messages. In W. A. Rosenbluth, editor, *Sensory Communication*, pages 217–234. MIT Press,

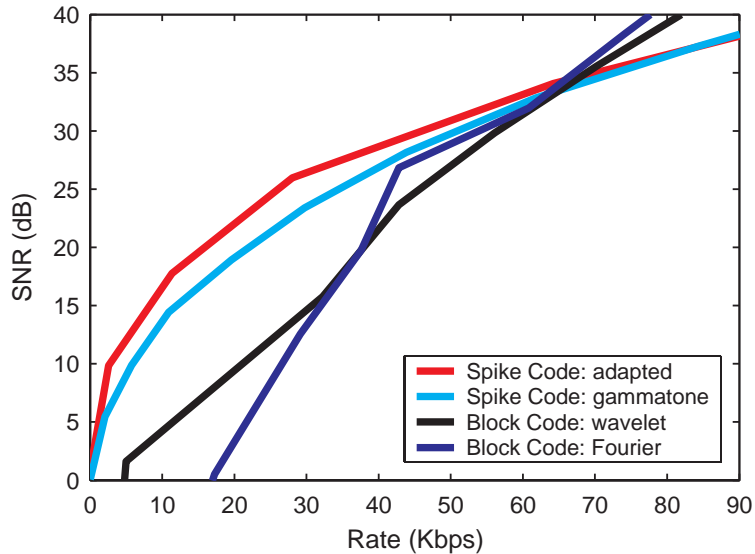

Figure 4: Rate-Fidelity curves speech were made for spike coding using both learned kernels (red) and gammatones (light blue) as well as using discrete Daubechies wavelet transform (black) and Fourier transform (dark blue).

Cambridge, 1961.

[2] J. J. Atick. Could information-theory provide an ecological theory of sensory processing. *Network*, 3(2):213–251, 1992.

[3] E. Simoncelli and B. Olshausen. Natural image statistics and neural representation. *Annual Review of Neuroscience*, 24:1193–1216, 2001.

[4] M. S. Lewicki. Efficient coding of natural sounds. *Nature Neuroscience*, 5(4):356–363, 2002.

[5] O. Schwartz and E. P. Simoncelli. Natural signal statistics and sensory gain control. *Nature Neuroscience*, 4:819–825, 2001.

[6] M. S. Lewicki. Efficient coding of time-varying patterns using a spiking population code. In R. P. N. Rao, B. A. Olshausen, and M. S. Lewicki, editors, *Probabilistic Models of the Brain: Perception and Neural Function*, pages 241–255. MIT Press, Cambridge, MA, 2002.

[7] E. C. Smith and M. S. Lewicki. Efficient coding of time-relative structure using spikes. *Neural Computation*, 2004.

[8] S. G. Mallat and Z. Zhang. Matching pursuits with time-frequency dictionaries. *IEEE Transactions on Signal Processing*, 41(12):3397–3415, 1993.

[9] B. A. Olshausen. Sparse codes and spikes. In R. P. N. Rao, B. A. Olshausen, and M. S. Lewicki, editors, *Probabilistic Models of the Brain: Perception and Neural Function*, pages 257–272. MIT Press, Cambridge, MA, 2002.

[10] L. H. Carney, M. J. McDuffy, and I. Shekhter. Frequency glides in the impulse responses of auditory-nerve fibers. *Journal of the Acoustical Society of America*, 105:2384–2391, 1999.

[11] J. S. Garofolo, L. F. Lamel, W. M. Fisher, J. G. Fiscus, D. S. Pallett, N. L. Dahlgren, and V. Zue. Timit acoustic-phonetic continuous speech corpus, 1990.
